# Probabilistic Modeling for Face Orientation Discrimination:
# Learning from Labeled and Unlabeled Data

**Shumeet Baluja**
baluja@cs.cmu.edu
Justsystem Pittsburgh Research Center &
School of Computer Science, Carnegie Mellon University

## Abstract

This paper presents probabilistic modeling methods to solve the problem of discriminating between five facial orientations with very little labeled data. Three models are explored. The first model maintains no inter-pixel dependencies, the second model is capable of modeling a set of arbitrary pair-wise dependencies, and the last model allows dependencies only between neighboring pixels. We show that for all three of these models, the accuracy of the learned models can be greatly improved by augmenting a small number of labeled training images with a large set of unlabeled images using Expectation-Maximization. This is important because it is often difficult to obtain image labels, while many unlabeled images are readily available. Through a large set of empirical tests, we examine the benefits of unlabeled data for each of the models. By using only two randomly selected labeled examples per class, we can discriminate between the five facial orientations with an accuracy of 94%; with six labeled examples, we achieve an accuracy of 98%.

## 1  Introduction

This paper examines probabilistic modeling techniques for discriminating between five face orientations: left profile, left semi-profile, frontal, right semi-profile, and right profile. Three models are explored: the first model represents no inter-pixel dependencies, the second model is capable of modeling a set of arbitrary pair-wise dependencies, and the last model allows dependencies only between neighboring pixels.

Models which capture inter-pixel dependencies can provide better classification performance than those that do not capture dependencies. The difficulty in using the more complex models, however, is that as more dependencies are modeled, more parameters must be estimated – which requires more training data. We show that by using Expectation-Maximization, the accuracy of what is learned can be greatly improved by augmenting a small number of labeled training images with unlabeled images, which are much easier to obtain.

The remainder of this section describes the problem of face orientation discrimination in detail. Section 2 provides a brief description of the probabilistic models explored. Section 3 presents results with these models with varying amounts of training data. Also shown is how Expectation-Maximization can be used to augment the limited labeled training data with unlabeled training data. Section 4 briefly discusses related work. Finally, Section 5 closes the paper with conclusions and suggestions for future work.

## 1.1 Detailed Problem Description

The interest in face orientation discrimination arises from two areas. First, the rapid increase in the availability of inexpensive cameras makes it practical to create systems which automatically monitor a person while using a computer. By using motion, color, and size cues, it is possible to quickly find and segment a person's face when he/she is sitting in front of a computer monitor. By determining whether the person is looking directly at the computer, or is staring away from the computer, we can provide feedback to any user interface that could benefit from knowing whether a user is paying attention or is distracted (such as computer-based tutoring systems for children, computer games, or even car-mounted cameras that monitor drivers).

Second, to perform accurate *face detection* for use in video-indexing or content-based image retrieval systems, one approach is to design detectors specific to each face orientation, such as [Rowley *et al.*, 1998, Sung 1996]. Rather than applying all detectors to every location, a face-orientation system can be applied to each candidate face location to "route" the candidate to the appropriate detector, thereby reducing the potential for false-positives, and also reducing the computational cost of applying each detector. This approach was taken in [Rowley *et al.*, 1998].

For the experiments in this paper, each image to be classified is 20x20 pixels. The face is centered in the image, and comprises most of the image. Sample faces are shown in Figure 1. Empirically, our experiments show that accurate pose discrimination is possible from binary versions of the images. First, the images were histogram-equalized to values between 0 and 255. This is a standard non-linear transformation that maps an approximately equal number of pixels to each value within the 0-255 range. It is used to improve the contrast in images. Second, to "binarize" the images, pixels with intensity above 128 were mapped to a value of 255, otherwise the pixels were mapped to a value of 0.

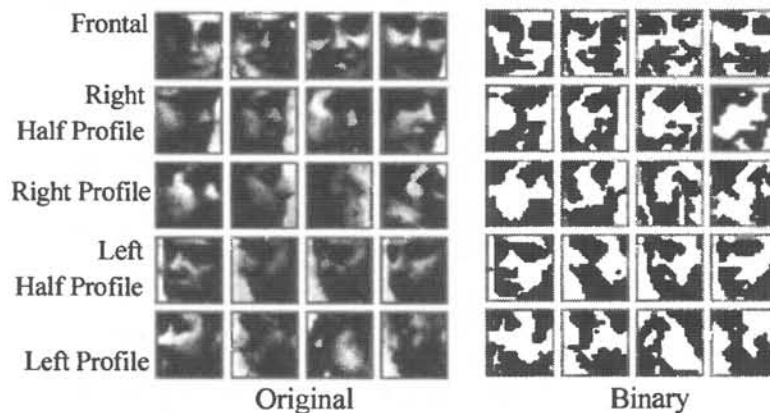

Frontal

Right
Half Profile

Right Profile

Left
Half Profile

Left Profile

Original                  Binary

**Figure 1:** 4 images of each of the 5 classes to be discriminated. Note the variability in the images. **Left:** Original Images. **Right:** Images after histogram equalization and binary quantization.

## 2  Methods Explored

This section provides a description of the probabilistic models explored: Naive-Bayes, Dependency Trees (as proposed by [Chow and Liu, 1968]), and a dependence network which models dependencies only between neighboring pixels. For more details on using Bayesian "multinets" (independent networks trained to model each class) for classification in a manner very similar to that used in this paper, see [Friedman, *et al.*, 1997].

### 2.1 The Naive-Bayes Model

The first, and simplest, model assumes that each pixel is independent of every other pixel. Although this assumption is clearly violated in real images, the model often yields good results with limited training data since it requires the estimation of the fewest parameters.

Assuming that each image belongs exclusively to one of the five face classes to be dis-

criminated, the probability of the image belonging to a particular class is given as follows:

$$P(Class_c|Image) = \frac{P(Image|Class_c) \times P(Class_c)}{P(Image)} \qquad P(Image|Class_c) = \prod_{i=1}^{400} P(Pixel_i|Class_c)$$

$P(Pixel_i|Class_c)$ is estimated directly from the training data by:

$$P(Pixel_i|Class_c) = \frac{k + \sum_{TrainingImages} Pixel_i \times P(Class_c|Image)}{2k + \sum_{TrainingImages} P(Class_c|Image)}$$

Since we are only counting examples from the training images, $P(Class_c|Image)$ is known. The notation $P(Class_c|Image)$ is used to represent image labels because it is convenient for describing the counting process with both labeled and unlabeled data (this will be described in detail in Section 3). With the labeled data, $P(Class_c|Image) \in \{0,1\}$. Later, $P(Class_c|Image)$ may not be binary; instead, the probability mass may be divided between classes. $Pixel_i \in \{0,1\}$ since the images are binary. $k$ is a smoothing constant, set to 0.001.

When used for classification, we compute the posterior probabilities and take the maximum, $C_{predicted}$, where: $C_{predicted} = \text{argmax}_c \ P(Class_c|Image) \cong P(Image|Class_c)$ . For simplicity, $P(Class_c)$ is assumed equal for all $c$; $P(Image)$ is a normalization constant which can be ignored since we are only interested in finding the maximum posterior probability.

## 2.2 Optimal Pair-Wise Dependency Trees

We wish to model a probability distribution $P(X_1, ..., X_{400}|Class_c)$, where each X corresponds to a pixel in the image. Instead of assuming pixel independence, we restrict our model to the following form:

$$P(X_1...X_n|Class_c) = \prod_{i=1}^{n} P\left(X_i|\Pi_{X_i}, Class_c\right)$$

where $\Pi_{X_i}$ is $X_i$'s single "parent" variable. We require that there be no cycles in these "parent-of" relationships: formally, there must exist some permutation $m = (m_1, ..., m_n)$ of $(1, ..., n)$ such that $\left(\Pi_{X_i} = X_j\right) \Rightarrow m(i) < m(j)$ for all $i$. In other words, we restrict P' to factorizations representable by Bayesian networks in which each node (except the root) has one parent, i.e., tree-shaped graphs.

A method for finding the optimal model within these restrictions is presented in [Chow and Liu, 1968]. A complete weighted graph G is created in which each variable $X_i$ is represented by a corresponding vertex $V_i$, and in which the weight $W_{ij}$ for the edge between vertices $V_i$ and $V_j$ is set to the mutual information $I(X_i,X_j)$ between $X_i$ and $X_j$. The edges in the maximum spanning tree of G determine an optimal set of (n-1) conditional probabilities with which to construct a tree-based model of the original probability distribution.

We calculate the probabilities $P(X_i)$ and $P(X_i, X_j)$ directly from the dataset. From these, we calculate the mutual information, $I(X_i, X_j)$, between all pairs of variables $X_i$ and $X_j$:

$$I(X_i, X_j) = \sum_{a,b} P(X_i = a, X_j = b) \cdot \log \frac{P(X_i = a, X_j = b)}{P(X_i = a) \cdot P(X_j = b)}$$

The maximum spanning tree minimizes the Kullback-Leibler divergence D(P||P') between

the true and estimated distributions:

$$D(P \| P') = \sum_X P(X) \log \frac{P(X)}{P'(X)}$$

as shown in [Chow & Liu, 1968]. Among all distributions of the same form, this distribution maximizes the likelihood of the data when the data is a set of empirical observations drawn from any unknown distribution.

## 2.3 Local Dependency Models

Unlike the Dependency Trees presented in the previous section, the local dependency networks only model dependencies between adjacent pixels. The most obvious dependencies to model are each pixel's eight neighbors. The dependencies are shown graphically in Figure 2(left). The difficulty with the above representation is that two pixels may be dependent upon each other (if this above model was represented as a Bayesian network, it would contain cycles). Therefore, to avoid problems with circular dependencies, we use the following model instead. Each pixel is still connected to each of its eight neighbors; however, the arcs are directed such that the dependencies are acyclic. In this local dependence network, each pixel is only dependent on four of its neighbors: the three neighbors to the right and the one immediately below. The dependencies which are modeled are shown graphically in Figure 2 (right). The dependencies are:

$$P(Image|Class_c) = \prod_{t=1}^{400} P(Pixel_t | \Pi_{P_{ixel_t}}, Class_c)$$

$$P(Pixel_{i,j}|\Pi_{P_{ixel_{i,j}}}, Class_c) = P(Pixel_{i,j}|Pixel_{i+1,j}, Pixel_{i+1,j+1}, Pixel_{i+1,j-1}, Pixel_{i,j+1}, Class_c)$$

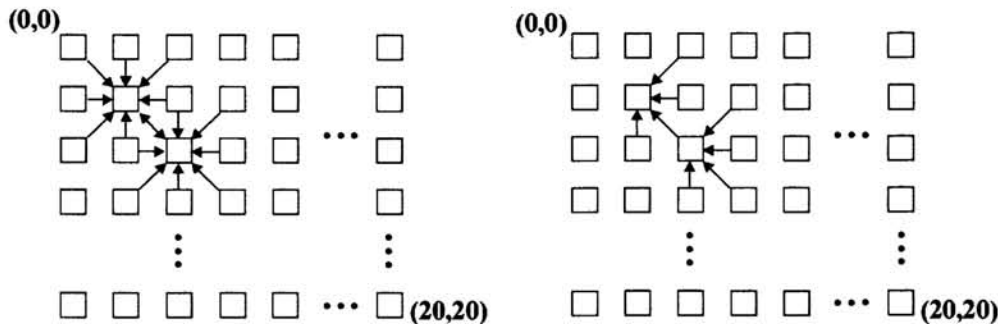

**Figure 2:** Diagram of the dependencies maintained. Each square represents a pixel in the image. Dependencies are shown only for two pixels. (Left) Model with 8 dependencies – note that because this model has circular dependencies, we do not use it. Instead, we use the model shown on the Right. (Right) Model used has 4 dependencies per pixel. By imposing an ordering on the pixels, circular dependencies are avoided.

## 3 Performance with Labeled and Unlabeled Data

In this section, we compare the results of the three probabilistic models with varying amounts of labeled training data. The training set consists of between 1 and 500 labeled training examples, and the testing set contains 5500 examples. Each experiment is repeated at least 20 times with random train/test splits of the data.

### 3.1 Using only Labeled Data

In this section, experiments are conducted with only labeled data. Figure 3(left) shows each model's accuracy in classifying the images in the test set into the five classes. As

expected, as more training data is used, the performance improves for all models.

Note that the model with no-dependencies performs the best when there is little data. However, as the amount of data increases, the relative performance of this model, compared to the other models which account for dependencies, decreases. It is interesting to note that when there is little data, the Dependency Trees perform poorly. Since these trees can select dependencies between any two pixels, they are the most susceptible to finding spurious dependencies. However, as the amount of data increases, the performance of this model rapidly improves. By using all of the labeled data (500 examples total), the Dependency Tree and the Local-Dependence network perform approximately the same, achieving a correct classification rate of approximately 99%.

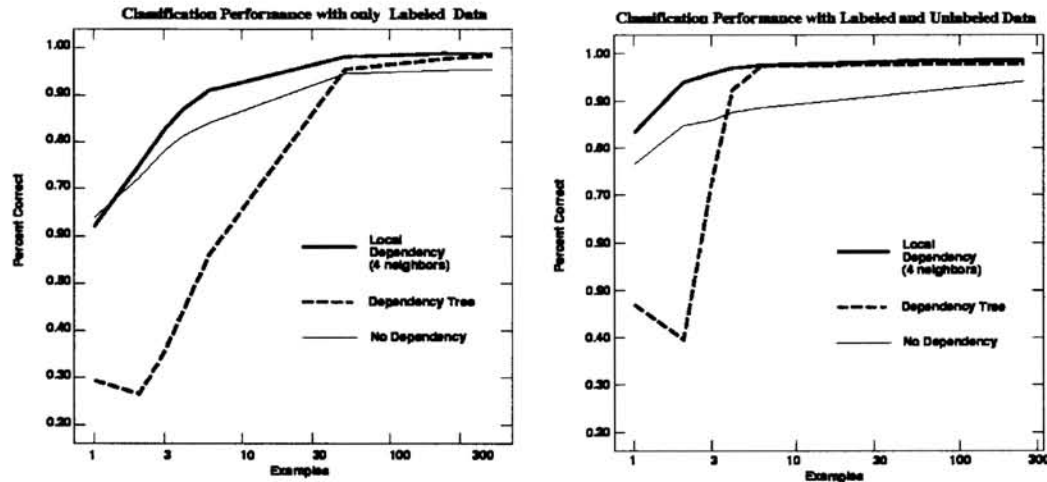

**Figure 3:** Performance of the three models. **X Axis:** Amount of labeled training data used. **Y Axis:** Percent correct on an independent test set. In the left graph, only labeled data was used. In the right graph, unlabeled and labeled data was used (the total number of examples were 500, with varying amounts of labeled data).

### 3.2 Augmenting the Models with Unlabeled Data

We can augment what is learned from only using the labeled examples by incorporating unlabeled examples through the use of the Expectation-Maximization (EM) algorithm. Although the details of EM are beyond the scope of this paper, the resulting algorithm is easily described (for a description of EM and applications to filling in missing values, see [Dempster *et al.*, 1977] and [Ghahramani & Jordan, 1994]):

1. Build the models using only the labeled data (as in Section 2).

2. Use the models to probabilistically label the unlabeled images.

3. Using the images with the probabilistically assigned labels, and the images with the given labels, recalculate the models' parameters. As mentioned in section 2, for the images labeled by this process, $P(Class_c|Image)$ is **not** restricted to $\{0,1\}$; the probability mass for an image may be spread to multiple classes.

4. If a pre-specified termination condition is not met, go to step 2.

This process is used for each classifier. The termination condition was five iterations; after five iterations, there was little change in the models' parameters.

The performance of the three classifiers with unlabeled data is shown in Figure 3(right). Note that with small amounts of data, the performance of all of the classifiers improved dramatically when the unlabeled data is used. Figure 4 shows the percent improvement by using the unlabeled data to augment the labeled data. Note that the error is reduced by

almost 90% with the use of unlabeled data (see the case with Dependency Trees with only 4 labeled examples, in which the accuracy rates increase from 44% to 92.5%). With only 50 labeled examples, a classification accuracy of 99% was obtained. This accuracy was obtained with almost an order of magnitude fewer labeled examples than required with classifiers which used only labeled examples.

In almost every case examined, the addition of unlabeled data helped performance. However, unlabeled data actually hurt the no-dependency model when a large amount of labeled data already existed. With large amounts of labeled data, the parameters of the model were estimated well. Incorporating unlabeled data may have hurt performance because the underlying generative process modeled did not match the real generative process. Therefore, the additional data provided may not have been labeled with the accuracy required to improve the model's classification performance. It is interesting to note that with the more complex models, such as the dependency trees or local dependence networks, even with the same amount of labeled data, unlabeled data improved performance. [Nigam, *et al.*, 1998] have reported similar performance degradation when using a large number of labeled examples and EM with a naive-Bayesian model to classify text documents. They describe two methods for overcoming this problem. First, they adjust the relative weight of the labeled and unlabeled data in the M-step by using cross-validation. Second, they providing multiple centroids per class, which improves the data/model fit. Although not presented here due to space limitations, the first method was attempted – it improved the performance on the face orientation discrimination task.

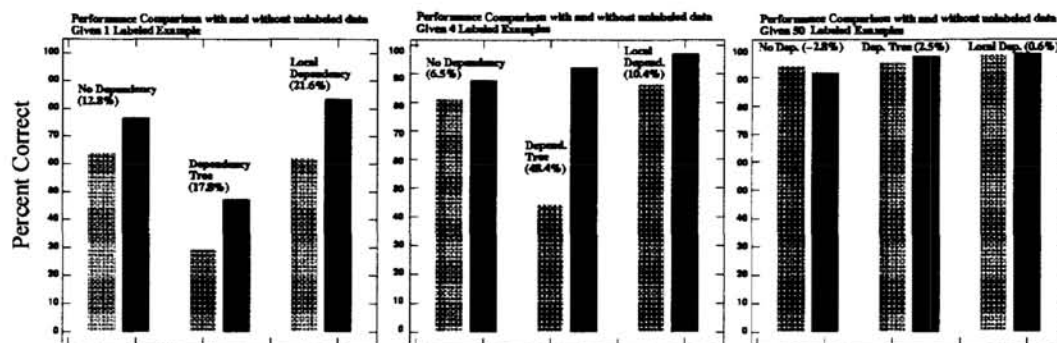

**Figure 4:** Improvement for each model by using unlabeled data to augment the labeled data. **Left:** with only 1 labeled example, **Middle:** 4 labeled, **Right:** 50 labeled. The bars in light gray represent the performance with only labeled data, the dark bars indicate the performance with the unlabeled data. The number in parentheses indicates the *absolute (in contrast to relative)* percentage change in classification performance with the use of unlabeled data.

## 4 Related Work

There is a large amount of work which attempts to discover attributes of faces, including (but not limited to) face detection, face expression discrimination, face recognition, and face orientation discrimination (for example [Rowley *et al.*, 1998][Sung, 1996][Bartlett & Sejnowski, 1997][Cottrell & Metcalfe, 1991][Turk & Pentland, 1991]). The work presented in this paper demonstrates the effective incorporation of unlabeled data into image classification procedures; it should be possible to use unlabeled data in any of these tasks.

The closest related work is presented in [Nigam *et al*, 1998]. They used naive-Bayes methods to classify text documents into a pre-specified number of groups. By using unlabeled data, they achieve significant classification performance improvement over using labeled documents alone. Other work which has employed EM for learning from labeled and unlabeled data include [Miller and Uyar, 1997] who used a mixture of experts classifier, and [Shahshahani & Landgrebe, 1994] who used a mixture of Gaussians. However, the dimensionality of their input was at least an order of magnitude smaller than used here. There is a wealth of other related work, such as [Ghahramani & Jordan, 1994] who have

used EM to fill in missing values in the training examples. In their work, class labels can be regarded as another feature value to fill-in.

Other approaches to reducing the need for large amounts of labeled data take the form of *active learning* in which the learner can ask for the labels of particular examples. [Cohn, *et. al* 1996] [McCallum & Nigam, 1998] provide good overviews of active learning.

## 5  Conclusions & Future Work

This paper has made two contributions. The first contribution is to solve the problem of discriminating between five face orientations with very little data. With only two labeled example images per class, we were able to obtain classification accuracies of 94% on separate test sets (with the local dependence networks with 4 parents). With only a few more examples, this was increased to greater than 98% accuracy. This task has a range of applications in the design of user-interfaces and user monitoring.

We also explored the use of multiple probabilistic models with unlabeled data. The models varied in their complexity, ranging from modeling no dependencies between pixels, to modeling four dependencies per pixel. While the no-dependency model performs well with very little labeled data, when given a large amount of labeled data, it is unable to match the performance of the other models presented. The Dependency-Tree models perform the worst when given small amounts of data because they are most susceptible to finding spurious dependencies in the data. The local dependency models performed the best overall, both by working well with little data, and by being able to exploit more data, whether labeled or unlabeled. By using EM to incorporate unlabeled data into the training of the classifiers, we improved the performance of the classifiers by up to approximately 90% when little labeled data was available.

The use of unlabeled data is vital in this domain. It is time-consuming to hand label many images, but many unlabeled images are often readily available. Because many similar tasks, such as face recognition and facial expression discrimination, suffer from the same problem of limited labeled data, we hope to apply the methods described in this paper to these applications. Preliminary results on related recognition tasks have been promising.

### Acknowledgments

Scott Davies helped tremendously with discussions about modeling dependencies. I would also like to acknowledge the help of Andrew McCallum for discussions of EM, unlabeled data and the related work. Many thanks are given to Henry Rowley who graciously provided the data set. Finally, thanks are given to Kaari Flagstad for comments on drafts of this paper.

### References

Bartlett, M. & Sejnowski, T. (1997) "Viewpoint Invariant Face Recognition using ICA and Attractor Networks", in *Adv. in Neural Information Processing Systems (NIPS) 9*.

Chow, C. & Liu, C. (1968) "Approximating Discrete Probability Distributions with Dependence Trees". *IEEE-Transactions on Information Theory*, 14: 462-467.

Cohn, D.A., Ghahramani, Z. & Jordan, M. (1996) "Active Learning with Statistical Models", *Journal of Artificial Intelligence Research* 4: 129-145.

Cottrell, G. & Metcalfe, (1991) "Face, Gender and Emotion Recognition using Holons", *NIPS 3*.

Dempster, A. P., Laird, N.M., Rubin, D.B. (1977) "Maximum Likelihood from Incomplete Data via the EM Algorithm", *J. Royal Statistical Society Series B*, 39 1-38.

Friedman, N., Geiger, D. Goldszmidt, M. (1997) "Bayesian Network Classifiers", *Machine Learning* 1:29.

Ghahramani & Jordan (1994) "Supervised Learning from Incomplete Data Via an EM Approach" *NIPS 6*.

McCallum, A. & Nigam, K. (1998) "Employing EM in Pool-Based Active Learning", in *ICML98*.

Miller, D. & Uyar, H. (1997) "A Mixture of Experts Classifier with Learning based on both Labeled and Unlabeled data", in *Adv. in Neural Information Processing Systems 9*.

Nigam, K. McCallum, A., Thrun, S., Mitchell, T. (1998), "Learning to Classify Text from Labeled and Unlabeled Examples", to appear in *AAAI-98*.

Rowley, H., Baluja, S. & Kanade, T. (1998) "Neural Network-Based Face Detection", *IEEE-Transactions on Pattern Analysis and Machine Intelligence (PAMI)*, Vol. 20, No. 1, January, 1998.

Shahshahani, B. & Landgrebe, D. (1994) "The Effect of Unlabeled samples in reducing the small sample size problem and mitigating the Hughes Phenomenon", *IEEE Trans. on Geosc. and Remote Sensing* 32.

Sung, K.K. (1996), *Learning and Example Selection for Object and Pattern Detection*, Ph.D. Thesis, MIT AI Lab - AI Memo 1572.

Turk, M. & Pentland, A. (1991) "Eigenfaces for Recognition". *J. Cog Neurosci*. 3 (1).